# A Weighted Probabilistic Neural Network

**David Montana**
Bolt Beranek and Newman Inc.
10 Moulton Street
Cambridge, MA 02138

## Abstract

The Probabilistic Neural Network (PNN) algorithm represents the likelihood function of a given class as the sum of identical, isotropic Gaussians. In practice, PNN is often an excellent pattern classifier, outperforming other classifiers including backpropagation. However, it is not robust with respect to affine transformations of feature space, and this can lead to poor performance on certain data. We have derived an extension of PNN called Weighted PNN (WPNN) which compensates for this flaw by allowing anisotropic Gaussians, i.e. Gaussians whose covariance is not a multiple of the identity matrix. The covariance is optimized using a genetic algorithm, some interesting features of which are its redundant, logarithmic encoding and large population size. Experimental results validate our claims.

## 1 INTRODUCTION

### 1.1 PROBABILISTIC NEURAL NETWORKS (PNN)

PNN (Specht 1990) is a pattern classification algorithm which falls into the broad class of "nearest-neighbor-like" algorithms. It is called a "neural network" because of its natural mapping onto a two-layer feedforward network. It works as follows. Let the exemplars from class $i$ be the $k$-vectors $\vec{x}_j^i$ for $j = 1, ..., N_i$. Then, the likelihood function for class $i$ is

$$L_i(\vec{x}) = \frac{1}{N_i(2\pi\sigma)^{k/2}} \sum_{j=1}^{N_i} e^{-(\vec{x}-\vec{x}_j^i)^2/\sigma} \tag{1}$$

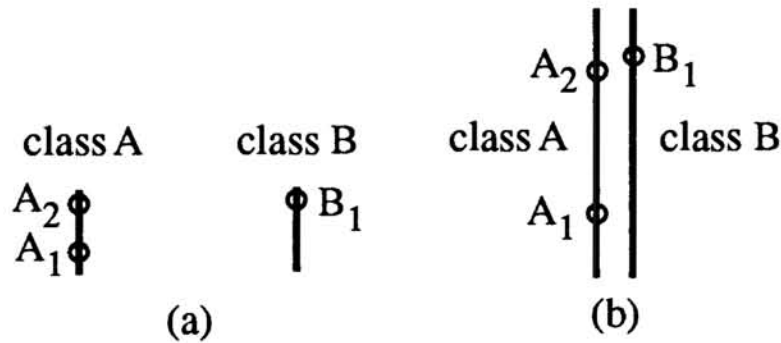

Figure 1: PNN is not robust with respect to affine transformations of feature space. Originally (a), $A_2$ is closer to its classmate $A_1$ than to $B_1$; however, after a simple affine transformation (b), $A_2$ is closer to $B_1$.

and the conditional probability for class $i$ is

$$P_i(\vec{x}) = L_i(\vec{x}) / \sum_{j=1}^{M} L_j(\vec{x}) \qquad (2)$$

Note that the class likelihood functions are sums of identical isotropic Gaussians centered at the exemplars.

The single free parameter of this algorithm is $\sigma$, the variance of the Gaussians (the rest of the terms in the likelihood functions are determined directly from the training data). Hence, training a PNN consists of optimizing $\sigma$ relative to some evaluation criterion, typically the number of classification errors during cross-validation (see Sections 2.1 and 3). Since the search space is one-dimensional, the search procedure is trivial and is often performed by hand.

## 1.2   THE PROBLEM WITH PNN

The main drawback of PNN and other "nearest-neighbor-like" algorithms is that they are not robust with respect to affine transformations (i.e., transformations of the form $\vec{x} \mapsto A\vec{x} + \vec{b}$) of feature space. (Note that in theory affine transformations should not affect the performance of backpropagation, but the results of Section 3 show that this is not true in practice.) Figures 1 and 2 depict examples of how affine transformations of feature space affect classification performance. In Figures 1a and 2a, the point $A_2$ is closer (using Euclidean distance) to point $A_1$, which is also from class A, than to point $B_1$, which is from class B. Hence, with a training set consisting of the exemplars $A_1$ and $B_1$, PNN would classify $A_2$ correctly. Figures 1b and 2b depict the feature space after affine transformations. In both cases, $A_2$ is closer to $B_1$ than to $A_1$ and would hence be classified incorrectly. For the example of Figure 2, the transformation matrix $A$ is not diagonal (i.e., the principle axes of the transformation are not the coordinate axes), and the adverse effects of this transformation cannot be undone by any affine transformation with diagonal $A$.

This problem has motivated us to generalize the PNN algorithm in such a way that it is robust with respect to affine transformations of the feature space.

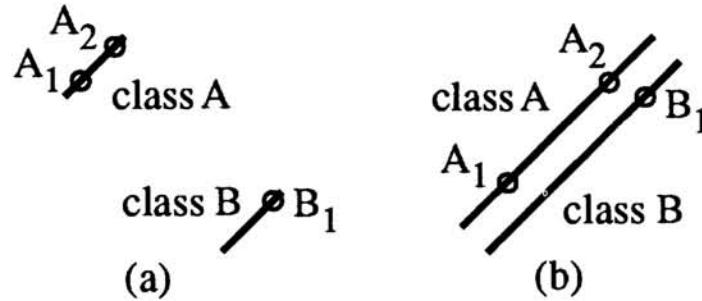

Figure 2: The principle axes of the affine transformation do not necessarily correspond with the coordinate axes.

## 1.3    A SOLUTION: WEIGHTED PNN (WPNN)

This flaw of nearest-neighbor-like algorithms has been recognized before, and there have been a few proposed solutions. They all use what Dasarathy (1991) calls "modified metrics", which are non-Euclidean distance measures in feature space. All the approaches to modified metrics define criteria which the chosen metric should optimize. Some criteria allow explicit derivation of the new metrics (Short and Fukunuga 1981; Fukunuga and Flick 1984). However, the validity of these derivations relies on there being a very large number of exemplars in the training set. A more recent set of approaches (Atkeson 1991; Kelly and Davis 1991) (i) use criteria which measure the performance on the training set using leaving-one-out cross-validation (see (Stone 1974) and Section 2.1), (ii) restrict the number of parameters of the metric to increase statistical significance, and (iii) optimize the parameters of the metric using non-linear search techniques. For his technique of "locally weighted regression", Atkeson (1991) uses an evaluation criterion which is the sum of the squares of the error using leaving-one-out. His metric has the form $d^2 = w_1(x_1 - y_1)^2 + ... + w_k(x_k - y_k)^2$, and hence has $k$ free parameters $w_1, ..., w_k$. He uses Levenberg-Marquardt to optimize these parameters with respect to the evaluation criterion. For their Weighted K-Nearest Neighbors (WKNN) algorithm, Kelly and Davis (1991) use an evaluation criterion which is the total number of incorrect classifications under leaving-one-out. Their metric is the same as Atkeson's, and their optmization is done with a genetic algorithm.

We use an approach similar to that of Atkeson (1991) and Kelly and Davis (1991) to make PNN more robust with respect to affine transformations. Our approach, called Weighted PNN (WPNN), works by using anisotropic Gaussians rather than the isotropic Gaussians used by PNN. An anisotropic Gaussian has the form $\frac{1}{(2\pi)^{k/2}(\det \Sigma)^{1/2}} e^{-(\vec{x} - \vec{x}_0)^T \Sigma^{-1} (\vec{x} - \vec{x}_0)}$. The covariance $\Sigma$ is a nonnegative-definite $k \times k$ symmetric matrix. Note that $\Sigma$ enters into the exponent of the Gaussian so as to define a new distance metric, and hence the use of anisotropic Gaussians to extend PNN is analogous to the use of modified metrics to extend other nearest-neighbor-like algorithms.

The likelihood function for class $i$ is

$$L_i(\vec{x}) = \frac{1}{N_i (2\pi)^{k/2} (\det \Sigma)^{1/2}} \sum_{j=1}^{N_i} e^{-(\vec{x} - \vec{x}_j^i)^T \Sigma^{-1} (\vec{x} - \vec{x}_j^i)} \tag{3}$$

and the conditional probability is still as given in Equation 2. Note that when $\Sigma$ is a multiple of the identity, i.e. $\Sigma = \sigma I$, Equation 3 reduces to Equation 1. Section 2 describes how we select the value of $\Sigma$.

*To ensure good generalization, we have so far restricted ourselves to diagonal covariances* (and thus metrics of the form used by Atkeson (1991) and Kelly and Davis (1991). This reduces the number of degrees of freedom of the covariance from $k(k+1)/2$ to $k$. However, this restricted set of covariances is not sufficiently general to solve all the problems of PNN (as demonstrated in Section 3), and we therefore in Section 2 hint at some modifications which would allow us to use arbitrary covariances.

## 2   OPTIMIZING THE COVARIANCE

We have used a genetic algorithm (Goldberg 1988) to optimize the covariance of the Gaussians. The code we used was a non-object-oriented C translation of the OOGA (Object-Oriented Genetic Algorithm) code (Davis 1991). This code preserves the features of OOGA including arbitrary encodings, exponential fitness, steady-state replacement, and adaptive operator probabilities. We now describe the distinguishing features of our genetic algorithm: (1) the evaluation function (Section 2.1), (2) the genetic encoding (Section 2.2), and (3) the population size (Section 2.3).

### 2.1   THE EVALUATION FUNCTION

To evaluate the performance of a particular covariance matrix on the training set, we use a technique called "leaving-one-out", which is a special form of cross-validation (Stone 1974). One exemplar at a time is withheld from the training set, and we then determine how well WPNN with that covariance matrix classifies the withheld exemplar. The full evaluation is the sum of the evaluations on the individual exemplars.

For the exemplar $\vec{x}_j^i$, let $\tilde{L}_q(\vec{x}_j^i)$ for $q = 1, ..., M$ denote the class likelihoods obtained upon withholding this exemplar and applying Equation 3, and let $\tilde{P}_q(\vec{x}_j^i)$ be the probabilities obtained from these likelihoods via Equation 2. Then, we define the performance as

$$E = \sum_{i=1}^{M} \sum_{j=1}^{N_i} ((1 - \tilde{P}_i(\vec{x}_j^i))^2 + \sum_{q \neq i} (\tilde{P}_q(\vec{x}_j^i))^2) \tag{4}$$

We have incorporated two heuristics to quickly identify covariances which are clearly bad and give them a value of $\infty$, the worst possible score. This greatly speeds up the optimization process because many of the generated covariances can be eliminated this way (see Section 2.3). The first heuristic identifies covariances which are too "small" based on the condition that, for some exemplar $\vec{x}_j^i$ and all $q = 1, ...M$, $\tilde{L}_q(\vec{x}_j^i) = 0$ to within the precision of IEEE double-precision floating-point format. In this case, the probabilities $\tilde{P}_q(\vec{x}_j^i)$ are not well-defined. (When $\Sigma$ is this "small", WPNN is approximately equivalent to WKNN with $k = 1$, and if such a small $\Sigma$ is indeed required, then the WKNN algorithm should be used instead.)

The second heuristic identifies covariances which are too "big" in the sense that too many exemplars contribute significantly to the likelihood functions. Empirical observations and theoretical arguments show that PNN (and WPNN) work best when only a small fraction of the exemplars contribute significantly. Hence, we reject a particular $\Sigma$ if, for any exemplar $\vec{x}_j^i$,

$$\sum_{\vec{x} \neq \vec{x}_j^i} e^{-(\vec{x}-\vec{x}_j^i)^T \Sigma^{-1} (\vec{x}-\vec{x}_j^i)} > (\sum_{i=1}^{M} N_i)/P \tag{5}$$

Here, $P$ is a parameter which we chose for our experiments to equal four.

**Note:** If we wish to improve the generalization by discarding some of the degrees of freedom of the covariance (which we will need to do when we allow non-diagonal covariances), we should modify the evaluation function by subtracting off a term which is montonically increasing with the number of degrees of freedom discarded.

## 2.2    THE GENETIC ENCODING

Recall from Section 1.3 that we have presently restricted the covariance to be diagonal. Hence, the set of all possible covariances is $k$-dimensional, where $k$ is the dimension of the feature space. We encode the covariances as $k+1$ integers $(a_0, ..., a_k)$, where the $a_i$'s are in the ranges $(a_0)_{min} \leq a_0 \leq (a_0)_{max}$ and $0 \leq a_i \leq a_{max}$ for $i = 1, ..., k$. The decoding map is

$$(a_0, ..., a_k) \mapsto \Sigma = \mathrm{diag}(2^{-(C_1 a_0 + C_2 a_1)}, ..., 2^{-(C_1 a_0 + C_2 a_k)}) \tag{6}$$

We observe the following about this encoding. First, it is a "logarithmic encoding", i.e. the encoded parameters are related logarithmically to the original parameters. This provides a large dynamic range without the sacrifice of sufficient resolution at any scale and without making the search space unmanageably large. The constants $C_1$ and $C_2$ determine the resolution, while the constants $(a_0)_{min}$, $(a_0)_{max}$, and $a_{max}$ determine the range. Second, it is possibly a "redundant" encoding, i.e. there may be multiple encodings of a single covariance. We use this redundant encoding, despite the seeming paradox, to reduce the size of the search space. The $a_0$ term encodes the size of the Gaussian, roughly equivalent to $\sigma$ in PNN. The other $a_i$'s encode the relative weighting of the various dimensions. If we dropped the $a_0$ term, the other $a_i$ terms would have to have larger ranges to compensate, thus making the search space larger.

**Note:** If we wish to improve the generalization by discarding some of the degrees of freedom of the covariance, we need to allow all the entries besides $a_0$ to take on the value of $\infty$ in addition to the range of values defined above. When $a_i = \infty$, its corresponding entry in the covariance matrix is zero and is hence discarded.

## 2.3    POPULATION SIZE

For their success, genetic algorithms rely on having multiple individuals with partial information in the population. The problem we have encountered is that the ratio of the the area of the search space with partial information to the entire search space is small. In fact, with our very loose heuristics, on Dataset 1 (see Section 3) about

90% of the randomly generated individuals of the initial population evaluated to $\infty$. In fact, we estimate very roughly that only 1 in 50 or 1 in 100 randomly generated individuals contain partial information. To ensure that the initial population has multiple individuals with partial information requires a population size of many hundreds, and we conservatively used a population size of 1600. Note that with such a large population it is essential to use a steady-state genetic algorithm (Davis 1991) rather than generational replacement.

## 3    EXPERIMENTAL RESULTS

We have performed a series of experiments to verify our claims about WPNN. To do so, we have constructed a sequence of four datasets designed to illustrate the shortcomings of PNN and how WPNN in its present form can fix some of these shortcomings but not others. Dataset 1 is a training set we generated during an effort to classify simulated sonar signals. It has ten features, five classes, and 516 total exemplars. Dataset 2 is the same as Dataset 1 except that we supplemented the ten features of Dataset 1 with five additional features, which were random numbers uniformly distributed between zero and one (and hence contained no information relevant to classification), thus giving a total of 15 features. Dataset 3 is the same as Dataset 2 except with ten (rather than five) irrelevant features added and hence a total of 20 features. Like Dataset 3, Dataset 4 has 20 features. It is obtained from Dataset 3 as follows. Pair each of the true features with one of the irrelevant features. Call the feature values of the $i^{th}$ pair $f_i$ and $g_i$. Then, replace these feature values with the values $0.5(f_i + g_i)$ and $0.5(f_i - g_i + 1)$, thus mixing up the relevant features with the irrelevant features via linear combinations.

To evaluate the performance of different pattern classification algorithms on these four datasets, we have used 10-fold cross-validation (Stone 1974). This involves splitting each dataset into ten disjoint subsets of similar size and similar distribution of exemplars by class. To evaluate a particular algorithm on a dataset requires ten training and test runs, where each subset is used as the test set for the algorithm trained on a training set consisting of the other nine subsets.

The pattern classification algorithms we have evaluated are backpropagation (with four hidden nodes), PNN (with $\sigma = 0.05$), WPNN and CART. The results of the experiments are shown in Figure 3. Note that the parenthesized quantities denote errors on the training data and are not compensated for the fact that each exemplar of the original dataset is in nine of the ten training sets used for cross-validation.

We can draw a number of conclusions from these results. First, the performance of PNN on Datasets 2-4 clearly demonstrates the problems which arise from its lack of robustness with respect to affine transformations of feature space. In each case, there exists an affine transformation which makes the problem essentially equivalent to Dataset 1 from the viewpoint of Euclidean distance, but the performance is clearly very different. Second, WPNN clearly eliminates this problem with PNN for Datasets 2 and 3 but not for Dataset 4. This points out both the progress we have made so far in using WPNN to make PNN more robust and the importance of extending the WPNN algorithm to allow non-diagonal covariances. Third, although backpropagation is in theory transparent to affine transformations of feature space (because the first layer of weights and biases implements an arbitrary affine

| Algorithm \ Dataset | 1 | 2 | 3 | 4 |
|---|---|---|---|---|
| Backprop | 11 (69) | 16 (51) | 20 (27) | 13 (64) |
| PNN | 9 | 94 | 109 | 29 |
| WPNN | 10 | 11 | 11 | 25 |
| CART | 14 | 17 | 18 | 53 |

Figure 3: Performance on the four datasets of backprop, CART, PNN and WPNN (parenthesized quantities are training set errors).

transformation), in practice affine transformations effect its performance. Indeed, Dataset 4 is obtained from Dataset 3 by an affine transformation, yet backpropagation performs very differently on them. Backpropagation does better on the training sets for Dataset 3 than on the training sets for Dataset 4 but does better on the test sets of Dataset 4 than the test sets of Dataset 3. This implies that for Dataset 4 during the training procedure backpropagation is not finding the globally optimum set of weights and biases but is missing in such a way that improves its generalization.

## 4    CONCLUSIONS AND FUTURE WORK

We have demonstrated through both theoretical arguments and experiments an inherent flaw of PNN, its lack or robustness with respect to affine transformations of feature space. To correct this flaw, we have proposed an extension of PNN, called WPNN, which uses anisotropic Gaussians rather than the isotropic Gaussians used by PNN. Under the assumption that the covariance of the Gaussians is diagonal, we have described how to use a genetic algorithm to optimize the covariance for optimal performance on the training set. Experiments have shown that WPNN can partially remedy the flaw with PNN.

What remains to be done is to modify the optimization procedure to allow arbitrary (i.e., non-diagonal) covariances. The main difficulty here is that the covariance matrix has a large number of degrees of freedom ($k(k+1)/2$, where $k$ is the dimension of feature space), and we therefore need to ensure that the choice of covariance is not overfit to the data. We have presented some general ideas on how to approach this problem, but a true solution still needs to be developed.

### Acknowledgements

This work was partially supported by DARPA via ONR under Contract N00014-89-C-0264 as part of the Artifical Neural Networks Initiative.

Thanks to Ken Theriault for his useful comments.

**References**

C.G. Atkeson. (1991) Using locally weighted regression for robot learning. *Proceedings of the 1991 IEEE Conference on Robotics and Automation*, pp. 958–963. Los Alamitos, CA: IEEE Computer Society Press.

B.V. Dasarathy. (1991) *Nearest Neighbor (NN) Norms: NN Pattern Classification Techniques*. Los Alamitos, CA: IEEE Computer Society Press.

L. Davis. (1991) *Handbook of Genetic Algorithms*. New York: Van Nostrand Reinhold.

K. Fukunaga and T.T. Flick. (1984) An optimal global nearest neighbor metric. *IEEE Transactions on Pattern Analysis and Machine Intelligence*, Vol. PAMI-6, No. 3, pp. 314–318.

D. Goldberg. (1988) *Genetic Algorithms in Machine Learning, Optimization and Search*. Redwood City, CA: Addison-Wesley.

J.D. Kelly, Jr. and L. Davis. (1991) Hybridizing the genetic algorithm and the k nearest neighbors classification algorithm. *Proceedings of the Fourth Internation Conference on Genetic Algorithms*, pp. 377–383. San Mateo, CA: Morgan Kaufmann.

R.D. Short and K. Fukunaga. (1981) The optimal distance measure for nearest neighbor classification. *IEEE Transactions on Information Theory*, Vol. IT-27, No. 5, pp. 622–627.

D.F. Specht. (1990) Probabilistic neural networks. *Neural Networks*, vol. 3, no. 1, pp. 109–118.

M. Stone. (1974) Cross-validatory choice and assessment of statistical predictions. *Journal of the Royal Statistical Society*, vol. 36, pp. 111–147.